# Relative Margin Machines

**Pannagadatta K Shivaswamy and Tony Jebara**
Department of Computer Science, Columbia University, New York, NY
pks2103,jebara@cs.columbia.edu

## Abstract

In classification problems, Support Vector Machines maximize the margin of separation between two classes. While the paradigm has been successful, the solution obtained by SVMs is dominated by the directions with large data spread and biased to separate the classes by cutting along large spread directions. This article proposes a novel formulation to overcome such sensitivity and maximizes the margin relative to the spread of the data. The proposed formulation can be efficiently solved and experiments on digit datasets show drastic performance improvements over SVMs.

## 1 Introduction

The goal of most machine learning problems is to generalize from a limited number of training examples. For example, in support vector machines [10] (SVMs) a hyperplane [1] of the form $\mathbf{w}^\top\mathbf{x} + b = 0$, $\mathbf{w} \in \mathbb{R}^m, \mathbf{x} \in \mathbb{R}^m, b \in \mathbb{R}$ is recovered as a decision boundary after observing a limited number of training examples. The parameters of the hyperplane $(\mathbf{w}, b)$ are estimated by maximizing the margin (the distance between $\mathbf{w}^\top\mathbf{x} + b = 1$ and $\mathbf{w}^\top\mathbf{x} + b = -1$) while minimizing a weighted upper bound on the misclassification rate on the training data (the so called slack variables). In practice, the margin is maximized by minimizing $\frac{1}{2}\mathbf{w}^\top\mathbf{w}$.

While this works well in practice, we point out that merely changing the scale of the data can give a different solution. On one hand, an adversary can exploit this shortcoming to transform the data so as to give bad performance. More distressingly, this shortcoming can naturally lead to a bad performance especially in high dimensional settings. The key problem is that SVMs simply find a large margin solution giving no attention to the spread of the data. An excellent discriminator lying in a dimension with relatively small data spread may be easily overlooked by the SVM solution. In this paper, we propose novel formulations to overcome such a limitation. The crux here is to find the maximum margin solution with respect to the spread of the data in a *relative* sense rather than finding the absolute large margin solution.

Linear discriminant analysis finds a projection of the data so that the inter-class separation is large while within class scatter is small. However, it only makes use of the first and the second order statistics of the data. Feature selection with SVMs [12] remove that have low discriminative value. Ellipsoidal kernel machines [9] normalize data in feature space by estimating bounding ellipsoids. While these previous methods showed performance improvements, both relied on multiple-step locally optimal algorithms for interleaving spread information with margin estimation. Recently, additional examples were used to improve the generalization of the SVMs with so called "Universum" samples [11]. Instead of leveraging additional data or additional model assumptions such as axis-aligned feature selection,

the proposed method overcomes what seems to be a fundamental limitation of the SVMs and subsequently yield improvements in the same supervised setting. In addition, the formulations derived in this paper are convex, can be efficiently solved and admit some useful generalization bounds.

**Notation** Boldface letters indicate vectors/matrices. For two vectors $\mathbf{u} \in \mathbb{R}^m$ and $\mathbf{v} \in \mathbb{R}^m$, $\mathbf{u} \leq \mathbf{v}$ indicates that $u_i \leq v_i$ for all $i$ from 1 to $m$. $\mathbf{1}$, $\mathbf{0}$ and $\mathbf{I}$ denote the vectors of all ones, all zeros and the identity matrix respectively. Their dimensions are clear from the context.

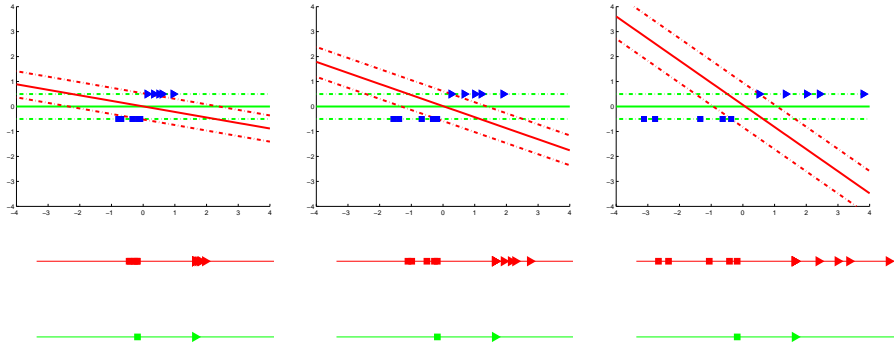

Figure 1: Top: As the data is scaled along the x-axis, the SVM solution (red or dark shade) deviates from the maximum relative margin solution (green or light shade). Bottom: The projections of the examples in the top row on the real line for the SVM solution (red or dark shade) and the proposed classifier (green or light shade) in each case.

## 2 Motivation with a two dimensional example

Let us start with a simple two dimensional toy dataset to illustrate a problem with the SVM solution. Consider the binary classification example shown in the top row of Figure 1 where squares denote examples from one class and triangles denote examples from the other class. Consider the leftmost plot in the top row of Figure 1. One possible decision boundary separating the two classes is shown in green (or light shade). The solution shown in red (or dark shade) is the SVM estimate; it achieves the largest margin possible while still separating both the classes. Is this necessarily "the best" solution?

Let us now consider the same set of points after scaling the x-axis in the second and the third plots. With progressive scaling, the SVM increasingly deviates from the green solution, clearly indicating that the SVM decision boundary is sensitive to affine transformations of the data and produces a family of different solutions as a result. This sensitivity to scaling and affine transformations is worrisome. If there is a best and a worst solution in the family of SVM estimates, there is always the possibility that an adversary exploits this scaling such that the SVM solution we recover is poor. Meanwhile, an algorithm producing the green decision boundary remains resilient to such adversarial scalings.

In the previous example, a direction with a small spread in the data produced a good discriminator. Merely finding a large margin solution, on the other hand, does not recover the best possible discriminator. This particular weakness in large margin estimation has only received limited attention in previous work. In the above example, suppose each class is generated from a one dimensional distribution on a line with the two classes on two parallel lines. In this case, the green decision boundary should obtain zero test error even if it is estimated from a finite number of samples. However, for finite training data, the SVM solution will make errors and will do so increasingly as the data is scaled along the x-axis. Using kernels and nonlinear mappings may help in some cases but might also exacerbate such problems. Similarly, simple prepossessing of the data (affine "whitening" to make the

dataset zero mean and unit covariance or scaling to place the data into a zero-one box) may fail to resolve such problems.

For more insight, consider the uni-dimensional projections of the data given by the green and red solutions in the bottom row of Figure 1. In the green solution, all points in the first class are mapped to a single coordinate and all points in the other class are mapped to another (distinct) coordinate. Meanwhile, the red solution produces more dispersed projections of the two classes. As the adversarial scaling is increased, the spread of the projection in the SVM solution increases correspondingly. Large margins are not sufficient on their own and what is needed is a way to also control the spread of the data after projection. Therefore, rather than just maximizing the margin, a trade-off regularizer should also be used to minimize the spread of the projected data. In other words, we will couple large margin estimation with regularization which seeks to bound the spread $|\mathbf{w}^\top \mathbf{x} + b|$ of the data. This will allow the linear classifier to recover large margin solutions not in the absolute sense but rather *relative to* the spread of the data in that projection direction.

## 3  Formulations

Given $(\mathbf{x}_i, y_i)_{i=1}^n$ where $\mathbf{x}_i \in \mathbb{R}^m$ and $y_i \in \{\pm 1\}$ drawn independent and identically distributed from a distribution $\Pr(\mathbf{x}, y)$, the Support Vector Machine primal formulation [2] is as follows:

$$\min_{\mathbf{w}, b, \boldsymbol{\xi} \geq 0} \frac{1}{2} \|\mathbf{w}\|^2 + C \boldsymbol{\xi}^\top \mathbf{1} \quad \text{s.t. } y_i(\mathbf{w}^\top \mathbf{x}_i + b) \geq 1 - \xi_i, \ \forall 1 \leq i \leq n. \tag{1}$$

The above formulation minimizes an upper bound on the misclassification while maximizing the margin (the two quantities are traded off by $C$). In practice, the following dual of the formulation (1) is solved:

$$\max_{0 \leq \boldsymbol{\alpha} \leq C\mathbf{1}} -\frac{1}{2} \sum_{i=1}^n \sum_{j=1}^n \alpha_i \alpha_j y_i y_j \mathbf{x}_i^\top \mathbf{x}_j + \sum_{i=1}^n \alpha_i \quad \text{s.t. } \boldsymbol{\alpha}^\top \mathbf{y} = 0. \tag{2}$$

It is easy to see that the above formulation (2) is rotation invariant; if all the $\mathbf{x}_i$ are replaced by $\mathbf{A}\mathbf{x}_i$ where $\mathbf{A} \in \mathbb{R}^{m \times m}, \mathbf{A}^\top \mathbf{A} = \mathbf{I}$, then the solution remains the same. However, the solution is not guaranteed to be the same when $\mathbf{A}$ is not a rotation matrix. In addition, the solution is sensitive to translations as well.

Typically, the dot product between the examples is replaced by a kernel function $k : \mathbb{R}^m \times \mathbb{R}^m \to \mathbb{R}$ such that $k(\mathbf{x}_i, \mathbf{x}_j) = \phi(\mathbf{x}_i)^\top \phi(\mathbf{x}_j)$, where $\phi : \mathbb{R}^m \to \mathcal{H}$ is a mapping to a Hilbert space to obtain non-linear decision boundaries in the input space. Thus, in (2), $\mathbf{x}_i^\top \mathbf{x}_j$ is replaced by $k(\mathbf{x}_i, \mathbf{x}_j)$ to obtain non-linear solutions. In rest of this paper, we denote by $\mathbf{K} \in \mathbb{R}^{n \times n}$ the Gram matrix, whose individual entries are given by $\mathbf{K}_{ij} = k(\mathbf{x}_i, \mathbf{x}_j)$.

Next, we consider the formulation which corresponds to whitening the data with the covariance matrix. Denote by $\boldsymbol{\Sigma} = \frac{1}{n} \sum_{i=1}^n \mathbf{x}_i \mathbf{x}_i^\top - \frac{1}{n^2} \sum_{i=1}^n \mathbf{x}_i \sum_{j=1}^n \mathbf{x}_j^\top$, and $\boldsymbol{\mu} = \frac{1}{n} \sum_{i=1}^n \mathbf{x}_i$, the sample covariance and mean respectively. Consider the following formulation which we call $\boldsymbol{\Sigma}$-SVM:

$$\min_{\mathbf{w}, b, \boldsymbol{\xi} \geq 0} \frac{1-D}{2} \|\mathbf{w}\|^2 + \frac{D}{2} \|\boldsymbol{\Sigma}^{\frac{1}{2}} \mathbf{w}\|^2 + C \boldsymbol{\xi}^\top \mathbf{1} \text{ s.t. } y_i(\mathbf{w}^\top (\mathbf{x}_i - \boldsymbol{\mu}) + b) \geq 1 - \xi_i, \tag{3}$$

where $0 \leq D \leq 1$ is an additional parameter that trades off between the two regularization terms.

The dual of (3) can be shown to be:

$$\max_{0 \leq \boldsymbol{\alpha} \leq C\mathbf{1}, \mathbf{y}^\top \boldsymbol{\alpha} = 0} \sum_{i=1}^n \alpha_i - \frac{1}{2} \sum_{i=1}^n \alpha_i y_i (\mathbf{x}_i - \boldsymbol{\mu})^\top ((1-D)\mathbf{I} + D\boldsymbol{\Sigma})^{-1} \sum_{j=1}^n \alpha_j y_j (\mathbf{x}_j - \boldsymbol{\mu}). \tag{4}$$

It is easy to see that the above formulation (4) is translation invariant and tends to an affine invariant solution when $D$ tends to one. When $0 < D < 1$, it can be shown, by using the Woodbury matrix inversion formula, that the above formulation can be "kernelized" simply by replacing the dot products $\mathbf{x}_i^\top \mathbf{x}_j$ in (2) by:

$$\frac{1}{1-D}\left(k(\mathbf{x}_i, \mathbf{x}_j) - \frac{\mathbf{K}_i^\top \mathbf{1}}{n} - \frac{\mathbf{K}_j^\top \mathbf{1}}{n} + \frac{\mathbf{1}^\top \mathbf{K} \mathbf{1}}{n^2}\right)$$

$$-\frac{1}{1-D}\left(\left(\mathbf{K}_i - \frac{\mathbf{K}\mathbf{1}}{n}\right)^\top \left(\frac{\mathbf{I}}{n} - \frac{\mathbf{1}\mathbf{1}^\top}{n^2}\right)\left[\frac{1-D}{D}\mathbf{I} + \mathbf{K}\left(\frac{\mathbf{I}}{n} - \frac{\mathbf{1}\mathbf{1}^\top}{n^2}\right)\right]^{-1}\left(\mathbf{K}_j - \frac{\mathbf{K}\mathbf{1}}{n}\right)\right),$$

where $\mathbf{K}_i$ is the $i^{th}$ column of $\mathbf{K}$. For $D = 0$ and $D = 1$, it is much easier to obtain the kernelized formulations. Note that the above formula involves a matrix inversion of size $n$, making the kernel computation alone $\mathcal{O}(n^3)$.

## 3.1 RMM and its geometrical interpretation

From Section 2, it is clear that large margin in the absolute sense might be deceptive and could merely be a by product of bad scaling of the data. To overcome this limitation, as we pointed out earlier, we need to bound the projections of the training examples as well. As in the two dimensional example, it is necessary to trade off between the margin and the spread of the data. We propose a slightly modified formulation in the next section that can be solved efficiently. For now, we write the following formulation, mainly to show how it compares with the $\mathbf{\Sigma}$-SVM. In addition, writing the dual of the following formulation gives some geometric intuition. Since we trade off between the projections and the margin, implicitly, we find large *relative* margin. Thus we call the following formulation the Relative Margin Machine (RMM):

$$\min_{\mathbf{w}, b, \boldsymbol{\xi} \geq 0} \frac{1}{2}\|\mathbf{w}\|^2 + C\boldsymbol{\xi}^\top \mathbf{1} \quad \text{s.t. } y_i(\mathbf{w}^\top \mathbf{x}_i + b) \geq 1 - \xi_i, \ \frac{1}{2}(\mathbf{w}^\top \mathbf{x}_i + b)^2 \leq \frac{B^2}{2}. \tag{5}$$

This is a quadratically constrained quadratic problem (QCQP). This formulation has one extra parameter $B$ in addition to the SVM parameter. Note that $B \geq 1$ since having a $B$ less than one would mean none of the examples would satisfy $y_i(\mathbf{w}^\top \mathbf{x}_i + b) \geq 1$. Let $\mathbf{w}_C$ and $b_C$ be the solutions obtained by solving the SVM (1) for a particular value of $C$, then $B > \max_i |\mathbf{w}_C^\top \mathbf{x}_i + b_C|$, makes the constraint on the second line in the formulation (5) inactive for each $i$ and the solution obtained is the same as the SVM estimate.

For smaller $B$ values, we start getting different solutions. Specifically, with a smaller $B$, we still find a large margin solution such that all the projections of the training examples are bounded by $B$. Thus by trying out different $B$ values, we explore different large margin solutions with respect to the projection and spread of the data.

In the following, we assume that the value of $B$ is smaller than the threshold mentioned above. The Lagrangian of (5) is given by:

$$\frac{1}{2}\|\mathbf{w}\|^2 + C\boldsymbol{\xi}^\top \mathbf{1} - \sum_{i=1}^n \alpha_i \left(y_i(\mathbf{w}^\top \mathbf{x}_i + b) - 1 + \xi_i\right) - \boldsymbol{\beta}^\top \boldsymbol{\xi} + \sum_{i=1}^n \lambda_i \left(\frac{1}{2}(\mathbf{w}^\top \mathbf{x}_i + b)^2 - \frac{1}{2}B^2\right),$$

where $\boldsymbol{\alpha}, \boldsymbol{\beta}, \boldsymbol{\lambda} \geq 0$ are the Lagrange multipliers corresponding to the constraints. Differentiating with respect to the primal variables and equating them to zero, it can be shown that:

$$(\mathbf{I} + \sum_{i=1}^n \lambda_i \mathbf{x}_i \mathbf{x}_i^\top)\mathbf{w} - b\sum_{i=1}^n \lambda_i \mathbf{x}_i = \sum_{i=1}^n \alpha_i y_i \mathbf{x}_i, \ \ b = \frac{1}{\boldsymbol{\lambda}^\top \mathbf{1}}(\sum_{i=1}^n \alpha_i y_i - \sum_{i=1}^n \lambda_i \mathbf{w}^\top \mathbf{x}_i), \ \ C\mathbf{1} = \boldsymbol{\alpha} + \boldsymbol{\beta}.$$

Denoting by $\mathbf{\Sigma}_{\boldsymbol{\lambda}} = \sum_{i=1}^n \lambda_i \mathbf{x}_i \mathbf{x}_i^\top - \frac{1}{\boldsymbol{\lambda}^\top \mathbf{1}} \sum_{i=1}^n \lambda_i \mathbf{x}_i \sum_{j=1}^n \lambda_j \mathbf{x}_j^\top$, and by $\boldsymbol{\mu}_{\boldsymbol{\lambda}} = \frac{1}{\boldsymbol{\lambda}^\top \mathbf{1}} \sum_{j=1}^n \lambda_j \mathbf{x}_j$ the dual of (5) can be shown to be:

$$\max_{0 \leq \boldsymbol{\alpha} \leq C\mathbf{1}, \boldsymbol{\lambda} \geq 0} \sum_{i=1}^n \alpha_i - \frac{1}{2}\sum_{i=1}^n \alpha_i y_i (\mathbf{x}_i - \boldsymbol{\mu}_{\boldsymbol{\lambda}})^\top (\mathbf{I} + \mathbf{\Sigma}_{\boldsymbol{\lambda}})^{-1} \sum_{j=1}^n \alpha_j y_j (\mathbf{x}_j - \boldsymbol{\mu}_{\boldsymbol{\lambda}}) - \frac{1}{2}B^2 \boldsymbol{\lambda}^\top \mathbf{1} \tag{6}$$

Note that the above formulation is translation invariant since $\boldsymbol{\mu_\lambda}$ is subtracted from each $\mathbf{x}_i$. $\boldsymbol{\Sigma_\lambda}$ corresponds to a "shape matrix" (potentially low rank) determined by $\mathbf{x}_i$'s that have non-zero $\lambda_i$. From the KKT conditions of (5), $\lambda_i(\frac{1}{2}(\mathbf{w}^\top\mathbf{x}_i + b)^2 - \frac{B^2}{2}) = 0$. Consequently $\lambda_i > 0$ implies $(\frac{1}{2}(\mathbf{w}^\top\mathbf{x}_i + b)^2 - \frac{B^2}{2}) = 0$.

Geometrically, in the above formulation (6), the data is whitened with the matrix $(\mathbf{I} + \boldsymbol{\Sigma_\lambda})$ while solving SVM. While this is similar to what is done by the $\boldsymbol{\Sigma}$-SVM, the matrix $(\mathbf{I} + \boldsymbol{\Sigma_\lambda})$ is determined jointly considering both the margin of the data and the spread. In contrast, in $\boldsymbol{\Sigma}$-SVM, whitening is simply a prepossessing step which can be done independently of the margin. Note that the constraint $\frac{1}{2}(\mathbf{w}^\top\mathbf{x}_i + b)^2 \leq \frac{1}{2}B^2$ can be relaxed with slack variables at the expense of one additional parameter however this will not be investigated in this paper.

The proposed formulation is of limited use unless it can be solved efficiently. Solving (6) amounts to solving a semi-definite program; it cannot scale beyond a few hundred data points. Thus, for efficient solution, we consider a different but equivalent formulation.

Note that the constraint $\frac{1}{2}(\mathbf{w}^\top\mathbf{x}_i + b)^2 \leq \frac{1}{2}B^2$ can be equivalently posed as two linear constraints : $(\mathbf{w}^\top\mathbf{x}_i + b) \leq B$ and $-(\mathbf{w}^\top\mathbf{x}_i + b) \leq B$. With these constraints replacing the quadratic constraint, we have a quadratic program to solve. In the primal, we have $4n$ constraints (including $\boldsymbol{\xi} \geq 0$ ) instead of the $2n$ constraints in the SVM. Thus, solving RMM as a standard QP has the same order of complexity as the SVM. In the next section, we briefly explain how the RMM can be solved efficiently from the dual.

### 3.2 Fast algorithm

The main idea for the fast algorithm is to have linear constraints bounding the projections rather than quadratic constraints. The fast algorithm that we developed is based on $SVM^{light}$ [5]. We first write the equivalent of (5) with linear constraints:

$$\min_{\mathbf{w},b,\boldsymbol{\xi}\geq 0} \frac{1}{2}\|\mathbf{w}\|^2 + C\boldsymbol{\xi}^\top\mathbf{1} \text{ s.t. } y_i(\mathbf{w}^\top\mathbf{x}_i + b) \geq 1 - \xi_i, \ \mathbf{w}^\top\mathbf{x}_i + b \leq B, \ -\mathbf{w}^\top\mathbf{x}_i - b \leq B. \quad (7)$$

The dual of (7) can be shown to be the following:

$$\max_{\boldsymbol{\alpha},\boldsymbol{\lambda},\boldsymbol{\lambda}^*} -\frac{1}{2}(\boldsymbol{\alpha}\otimes\mathbf{y} - \boldsymbol{\lambda} + \boldsymbol{\lambda}^*)^\top \mathbf{K}(\boldsymbol{\alpha}\otimes\mathbf{y} - \boldsymbol{\lambda} + \boldsymbol{\lambda}^*) + \boldsymbol{\alpha}^\top\mathbf{1} - B\boldsymbol{\lambda}^\top\mathbf{1} - B\boldsymbol{\lambda}^{*\top}\mathbf{1} \quad (8)$$

$$\text{s.t. } \boldsymbol{\alpha}^\top\mathbf{y} - \boldsymbol{\lambda}^\top\mathbf{1} + \boldsymbol{\lambda}^{*\top}\mathbf{1} = 0, \ 0 \leq \boldsymbol{\alpha} \leq C\mathbf{1}, \ \boldsymbol{\lambda},\boldsymbol{\lambda}^* \geq \mathbf{0},$$

where, the operator $\otimes$ denotes the element-wise product of two vectors.

The above QP (8) is solved in an iterative way. In each step, only a subset of the dual variables are optimized. Let us say, $q$, $r$ and $s$ ($\tilde{q}$, $\tilde{r}$ and $\tilde{s}$) are the indices to the free (fixed) variables in $\boldsymbol{\alpha}$, $\boldsymbol{\lambda}$ and $\boldsymbol{\lambda}^*$ respectively (such that $q \cup \tilde{q} = \{1, 2, \cdots n\}$ and $q \cap \tilde{q} = \emptyset$, similarly for the other two indices) in a particular iteration. Then the optimization over the free variables in that step can be expressed as:

$$\max_{\boldsymbol{\alpha}_q,\boldsymbol{\lambda}_r,\boldsymbol{\lambda}_s^*} -\frac{1}{2}\begin{bmatrix} \boldsymbol{\alpha}_q\otimes\mathbf{y}_q \\ \boldsymbol{\lambda}_r \\ \boldsymbol{\lambda}_s^* \end{bmatrix}^\top \begin{bmatrix} \mathbf{K}_{qq} & -\mathbf{K}_{qr} & \mathbf{K}_{qs} \\ -\mathbf{K}_{rq} & \mathbf{K}_{rr} & -\mathbf{K}_{rs} \\ \mathbf{K}_{sq} & -\mathbf{K}_{sr} & \mathbf{K}_{ss} \end{bmatrix} \begin{bmatrix} \boldsymbol{\alpha}_q\otimes\mathbf{y}_q \\ \boldsymbol{\lambda}_r \\ \boldsymbol{\lambda}_s^* \end{bmatrix} \quad (9)$$

$$-\frac{1}{2}\begin{bmatrix} \boldsymbol{\alpha}_q\otimes\mathbf{y}_q \\ \boldsymbol{\lambda}_r \\ \boldsymbol{\lambda}_s^* \end{bmatrix}^\top \begin{bmatrix} \mathbf{K}_{q\tilde{q}} & -\mathbf{K}_{q\tilde{r}} & \mathbf{K}_{q\tilde{s}} \\ -\mathbf{K}_{r\tilde{q}} & \mathbf{K}_{r\tilde{r}} & -\mathbf{K}_{r\tilde{s}} \\ \mathbf{K}_{s\tilde{q}} & -\mathbf{K}_{s\tilde{r}} & \mathbf{K}_{s\tilde{s}} \end{bmatrix} \begin{bmatrix} \boldsymbol{\alpha}_{\tilde{q}}\otimes\mathbf{y}_{\tilde{q}} \\ \boldsymbol{\lambda}_{\tilde{r}} \\ \boldsymbol{\lambda}_{\tilde{s}}^* \end{bmatrix} + \boldsymbol{\alpha}_q^\top\mathbf{1} - B\boldsymbol{\lambda}_r^\top\mathbf{1} - B\boldsymbol{\lambda}_s^{*\top}\mathbf{1}$$

$$\text{s.t. } \boldsymbol{\alpha}_q^\top y_q - \boldsymbol{\lambda}_r^*\mathbf{1} + \boldsymbol{\lambda}_s^{*\top}\mathbf{1} = -\boldsymbol{\alpha}_{\tilde{q}}^\top y_{\tilde{q}} + \boldsymbol{\lambda}_{\tilde{r}}^\top\mathbf{1} - \boldsymbol{\lambda}_{\tilde{s}}^{*\top}\mathbf{1}, \ \ 0 \leq \boldsymbol{\alpha}_q \leq C\mathbf{1}, \ \ \boldsymbol{\lambda}_r, \ \boldsymbol{\lambda}_s^* \geq \mathbf{0}.$$

Note that while the first term in the objective above is quadratic in the free variables (over which it is optimized), the second term is only linear.

The algorithm, solves a small sub-problem like (9) in each step until the KKT conditions of the formulation (8) are satisfied to a given tolerance. In each step, the free variables are selected using heuristics similar to those in $SVM^{light}$ but slightly adapted to our formulation.

We omit the details due to lack of space. Since only a small subset of the variables is optimized, book-keeping can be done efficiently in each step. Moreover, the algorithm can be warm-started with a previous solution just like $SVM^{light}$.

## 4  Experiments

Experiments were carried out on three sets of digits - optical digits from the UCI machine learning repository [1], USPS digits [6] and MNIST digits [7]. These datasets have different number of features (64 in optical digits, 256 in USPS and 784 in MNIST) and training examples (3823 in optical digits, 7291 in USPS and 60000 in MNIST). In all these multi-class experiments one versus one classification strategy was used. We start by noting that, on the MNIST test set, an improvement of 0.1% is statistically significant [3, 4]. This corresponds to 10 or fewer errors by one method over another on the MNIST test set.

All the parameters were tuned by splitting the training data in each case in the ratio 80:20 and using the smaller split for validation and the larger split for training. The process was repeated five times over random splits to pick best parameters ($C$ for SVM, $C$ and $D$ for $\mathbf{\Sigma}$-SVM and $C$ and $B$ for RMM). A final classifier was trained for each of the 45 classification problems with the best parameters found from cross validation using all the training examples in those classes.

In the case of MNIST digits, training $\mathbf{\Sigma}$-SVM and KLDA are prohibitive since they involve inverting a matrix. So, to compare all the methods, we conducted an experiment with 1000 examples per training. For the larger experiments we simply excluded $\mathbf{\Sigma}$-SVM and KLDA. The larger experiment on MNIST consisted of training with two thirds of the digits (note that this amounts to training with 8000 examples on an average for each pair of digits) for each binary classification task. In both the experiments, the remaining training data was used as a validation set. The classifier that performed the best on the validation set was used for testing.

Once we had 45 classifiers for each pair of digits, testing was done on the separate test set available in each of these three datasets (1797 examples in the case of optical digits, 2007 examples in USPS and 10000 examples in MNIST). The final prediction given for each test example was based on the majority of predictions made by the 45 classifiers on the test example with ties broken uniformly at random.

Table 1 shows the result on all the three datasets for polynomial kernel with various degrees and the RBF kernel. For each dataset, we report the number of misclassified examples using the majority voting scheme mentioned above. It can be seen that while $\mathbf{\Sigma}$-SVM usually performs much better compared to SVM, RMM performs even better than $\mathbf{\Sigma}$-SVM in most cases. Interestingly, with higher degree kernels, $\mathbf{\Sigma}$-SVM seems to match the performance of the RMM, but in most of the lower degree kernels, RMM outperforms both SVM and $\mathbf{\Sigma}$-SVM convincingly. Since, $\mathbf{\Sigma}$-SVM is prohibitive to run on large scale datasets, the RMM was clearly the most competitive method in these experiments.

**Training with entire MNIST**  We used the best parameters found by crossvalidation in the previous experiments on MNIST and trained 45 classifiers for both SVM and RMM with *all* the training examples for each class in MNIST for various kernels. The test results are reported in Table 1; the advantage still carries over to the full MNIST dataset.

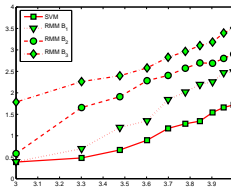

Figure 2: Log run time versus log number of examples from 1000 to 10000 in steps of 1000.

| | | 1 | 2 | 3 | 4 | 5 | 6 | 7 | RBF |
|---|---|---|---|---|---|---|---|---|---|
| OPT | SVM | 71 | 57 | 54 | 47 | 40 | 46 | 46 | 51 |
| | Σ-SVM | **61** | 48 | 41 | 36 | 35 | 31 | **29** | 47 |
| | KLDA | 71 | 57 | 54 | 47 | 40 | 46 | 46 | **45** |
| | RMM | 71 | **36** | **32** | **31** | **33** | **30** | **29** | 51 |
| USPS | SVM | 145 | 109 | 109 | 103 | 100 | 95 | 93 | 104 |
| | Σ-SVM | **132** | **108** | 99 | 94 | **89** | **87** | 90 | **97** |
| | KLDA | 132 | 119 | 121 | 117 | 114 | 118 | 117 | 101 |
| | RMM | 153 | 109 | **94** | **91** | 91 | 90 | **90** | 98 |
| 1000-MNIST | SVM | 696 | 511 | 422 | 380 | 362 | 338 | 332 | 670 |
| | Σ-SVM | **671** | 470 | 373 | 341 | 322 | 309 | 303 | 673 |
| | KLDA | 1663 | 848 | 591 | 481 | 430 | 419 | 405 | 1597 |
| | RMM | 689 | **342** | **319** | **301** | **298** | **290** | **296** | **613** |
| 2/3-MNIST | SVM | 552 | 237 | 200 | 183 | 178 | 177 | 164 | 166 |
| | RMM | **534** | **164** | **148** | **140** | **123** | **129** | **129** | **144** |
| Full MNIST | SVM | 536 | 198 | 170 | 156 | 157 | 141 | 136 | 146 |
| | RMM | **521** | **146** | **140** | **130** | **119** | **116** | **115** | **129** |

Table 1: Number of digits misclassified with various kernels by SVM, Σ-SVM and RMM for three different datasets.

**Run time comparison** We studied the empirical run times using the MNIST digits 3 vs 8 and polynomial kernel with degree two. The tolerance was set to 0.001 in both the cases. The size of the sub-problem (9) solved was 500 in all the cases. The number of training examples were increased in steps of 1000 and the training time was noted. $C$ value was set at 1000. SVM was first run on the training examples. The value of maximum absolute prediction $\theta$ was noted. We then tried three different values of $B$ for RMM, $B_1 = 1 + (\theta - 1)/2$, $B_2 = 1 + (\theta - 1)/4$ $B_3 = 1 + (\theta - 1)/10$. In all the cases, the run time was noted. We show a log-log plot comparing the number of examples to the run time in Figure 2. Both SVM and RMM have similar asymptotic behavior. However, in many cases, warm starting RMM with previous solution significantly helped in reducing the run times.

## 5   Conclusions

We identified a sensitivity of Support Vector Machines and maximum absolute margin criteria to affine scalings. These classifiers are biased towards producing decision boundaries that separate data along directions with large data spread. The Relative Margin Machine was proposed to overcome such a problem and optimizes the projection direction such that the margin is large only *relative to* the spread of the data. By deriving the dual with quadratic constraints, a geometrical interpretation was also formulated for RMMs. An implementation for RMMs requiring only additional linear constraints in the SVM quadratic program leads to a competitively fast implementation. Experiments showed that while affine transformations can improve over the SVMs, RMM performs even better in practice.

The maximization of relative margin is fairly promising as it is compatible with other popular problems handled by the SVM framework such as ordinal regression, structured prediction etc. These are valuable future extensions for the RMM. Furthermore, the constraints that bound the projection are unsupervised; thus RMMs can readily work in semi-supervised and transduction problems. We will study these extensions in detail in an extended version of this paper.

## Footnotes

[1]In this paper we use the dot product $\mathbf{w}^\top\mathbf{x}$ with the understanding that it can be replaced with an inner product.

[2]After this formulation, we stop explicitly writing $\forall 1 \leq i \leq n$ since it will be obvious from the context.

## References

[1] A. Asuncion and D.J. Newman. UCI machine learning repository, 2007.

[2] P. L. Bartlett and S. Mendelson. Rademacher and Gaussian complexities: Risk bounds and structural results. *Journal of Machine Learning Research*, 3:463–482, 2002.

[3] Y. Bengio, P. Lamblin, D. Popovici, and H. Larochelle. Greedy layer-wise training of deep networks. In *Advances in Neural Information Processing Systems 19*, pages 153–160. MIT Press, Cambridge, MA, 2007.

[4] D. Decoste and B. Schölkopf. Training invariant support vector machines. *Machine Learning*, pages 161–190, 2002.

[5] T. Joachims. Making large-scale support vector machine learning practical. In *Advances in Kernel Methods: Support Vector Machines*. MIT Press, Cambridge, MA, 1998.

[6] Y. LeCun, B. Boser, J.S. Denker, D. Henderson, R.E. Howard, W. Hubbard, and L. Jackel. Back-propagation applied to handwritten zip code recognition. *Neural Computation*, 1:541–551, 1989.

[7] Y. LeCun, L. Bottou, Y. Bengio, and P. Haffner. Gradient-based learning applied to document recognition. *Proceedings of the IEEE*, 86(11):2278–2324, 1998.

[8] J. Shawe-Taylor and N. Cristianini. *Kernel Methods for Pattern Analysis*. Cambridge University Press, 2004.

[9] P. K. Shivaswamy and T. Jebara. Ellipsoidal kernel machines. In *Proceedings of the Artificial Intelligence and Statistics*, 2007.

[10] V. Vapnik. *The Nature of Statistical Learning Theory*. Springer Verlag, New York, 1995.

[11] J. Weston, R. Collobert, F. H. Sinz, L. Bottou, and V. Vapnik. Inference with the universum. In *Proceedings of the International Conference on Machine Learning*, pages 1009–1016, 2006.

[12] J. Weston, S. Mukherjee, O. Chapelle, M. Pontil, T. Poggio, and V. Vapnik. Feature selection for SVMs. In *Neural Information Processing Systems*, pages 668–674, 2000.

## A   Generalization Bound

In this section, we give the empirical Rademacher complexity [2, 8] for function classes used by the SVM, and modified versions of RMM and $\mathbf{\Sigma}$-SVM which can be plugged into a generalization bound.

Maximizing the margin can be seen as choosing a function $f(\mathbf{x}) = \mathbf{w}^\top \mathbf{x}$ from a bounded class of functions $\mathcal{F}_E := \{\mathbf{x} \to \mathbf{w}^\top \mathbf{x} | \frac{1}{2}\|\mathbf{w}\|^2 \leq E\}$. For a technical reason, instead of bounding the projection on the training examples as in (5), we consider bounding the projections on an independent set of examples drawn from $\Pr(\mathbf{x})$, that is, a set $U = \{\mathbf{u}_1, \mathbf{u}_2, \ldots \mathbf{u}_{n_u}\}$. Note that if we have an *iid* training set, it can be split into two parts and one part can be used exclusively to bound the projections and the other part can be used exclusively for classification constraints. Since the labels of the examples used to bound the projections do not matter, we denote this set by $U$ and the other part of the set by $(\mathbf{x}_i, y_i)_{i=1}^n$ We now consider the following function class which is closely related to RMM: $\mathcal{H}_{E,D} := \{\mathbf{x} \to \mathbf{w}^\top \mathbf{x} | \frac{1}{2}\mathbf{w}^\top \mathbf{w} + \frac{D}{2}(\mathbf{w}^\top \mathbf{u}_i)^2 \leq E \ \forall 1 \leq i \leq n_u\}$ where $D > 0$ trades off between large margin and small bound on the projections. Similarly, consider: $\mathcal{G}_{E,D} := \{\mathbf{x} \to \mathbf{w}^\top \mathbf{x} | \frac{1}{2}\mathbf{w}^\top \mathbf{w} + \frac{D}{2n_u}\sum_{i=1}^{n_u}(\mathbf{w}^\top \mathbf{u}_i)^2 \leq E\}$, which is closely related to the class of functions considered by $\mathbf{\Sigma}$-SVM. The empirical Rademacher complexities of the three classes of functions are as below:

$$\hat{R}(\mathcal{F}_E) \leq U_{\mathcal{F}_E} := \frac{2\sqrt{2E}}{n}\sqrt{\sum_{i=1}^n \mathbf{x}_i^\top \mathbf{x}_i}, \qquad \hat{R}(\mathcal{G}_{E,D}) \leq U_{\mathcal{G}_{E,D}} := \frac{2\sqrt{2E}}{n}\sqrt{\sum_{i=1}^n \mathbf{x}_i^\top \mathbf{\Sigma}_D^{-1} \mathbf{x}_i},$$

$$\hat{R}(\mathcal{H}_{E,D}) \leq U_{\mathcal{H}_{E,D}} := \min_{\boldsymbol{\lambda} \geq 0} \frac{1}{n}\sum_{i=1}^n \mathbf{x}_i^\top \mathbf{\Sigma}_{\boldsymbol{\lambda},D}^{-1} \mathbf{x}_i + \frac{2}{n}E\sum_{i=1}^{n_u}\lambda_i,$$

where $\mathbf{\Sigma}_D = \mathbf{I} + \frac{D}{n_u}\sum_{i=1}^{n_u}\mathbf{u}_i\mathbf{u}_i^\top$ and $\mathbf{\Sigma}_{\boldsymbol{\lambda},D} = \sum_{i=1}^{n_u}\lambda_i\mathbf{I} + D\sum_{i=1}^{n_u}\lambda_i\mathbf{u}_i\mathbf{u}_i^\top$. Note that the last upper bound is not a closed form expression, but a semi-definite optimization. Now, the upper bounds $U_{\mathcal{F}_E}$, $U_{\mathcal{G}_{E,D}}$ and $U_{\mathcal{H}_{E,D}}$ can be plugged in the following theorem in place of $\hat{R}(\mathcal{F})$ to obtain Rademacher type generalization bounds.

**Theorem 1** *Fix $\gamma > 0$, let $\mathcal{F}$ be the class of functions from $\mathbb{R}^m \times \{\pm 1\} \to \mathbb{R}$ given by $f(\mathbf{x}, y) = -yg(\mathbf{x})$. Let $\{(\mathbf{x}_1, y_1), \ldots, (\mathbf{x}_n, y_n)\}$ be drawn iid from a probability distribution $\mathcal{D}$. Then, with probability at least $1 - \delta$ over the samples of size $n$, the following bound holds: $\Pr_{\mathcal{D}}[y \neq sign(g(\mathbf{x}))] \leq \boldsymbol{\xi}^\top \mathbf{1}/n + 2\hat{R}(\mathcal{F})/\gamma + 3\sqrt{(\ln(2/\delta))/2n}$, where $\xi_i = \max(0, 1 - y_i g(\mathbf{x}_i))$ are the so-called slack variables.*

